# A New Probabilistic Model for Rank Aggregation

**Tao Qin**
Microsoft Research Asia
taoqin@microsoft.com

**Xiubo Geng**
Chinese Academy of Sciences
xiubogeng@gmail.com

**Tie-Yan Liu**
Microsoft Research Asia
tyliu@microsoft.com

## Abstract

This paper is concerned with rank aggregation, which aims to combine multiple input rankings to get a better ranking. A popular approach to rank aggregation is based on probabilistic models on permutations, e.g., the Luce model and the Mallows model. However, these models have their limitations in either poor expressiveness or high computational complexity. To avoid these limitations, in this paper, we propose a new model, which is defined with a coset-permutation distance, and models the generation of a permutation as a stagewise process. We refer to the new model as coset-permutation distance based stagewise (CPS) model. The CPS model has rich expressiveness and can therefore be used in versatile applications, because many different permutation distances can be used to induce the coset-permutation distance. The complexity of the CPS model is low because of the stagewise decomposition of the permutation probability and the efficient computation of most coset-permutation distances. We apply the CPS model to supervised rank aggregation, derive the learning and inference algorithms, and empirically study their effectiveness and efficiency. Experiments on public datasets show that the derived algorithms based on the CPS model can achieve state-of-the-art ranking accuracy, and are much more efficient than previous algorithms.

## 1 Introduction

Rank aggregation aims at combining multiple rankings of objects to generate a better ranking. It is the key problem in many applications. For example, in meta search [1], when users issue a query, the query is sent to several search engines and the rankings given by them are aggregated to generate more comprehensive ranking results.

Given the underlying correspondence between ranking and permutation, probabilistic models on permutations, originated in statistics [19, 5, 4], have been widely applied to solve the problems of rank aggregation. Among different models, the Mallows model [15, 6] and the Luce model [14, 18] are the most popular ones.

The *Mallows model* is a distance-based model, which defines the probability of a permutation according to its distance to a location permutation. Due to many applicable permutation distances, the Mallows model has very rich expressiveness, and therefore can be potentially used in many different applications. Its weakness lies in the high computational complexity. In many cases, it requires a time complexity of $O(n!)$ to compute the probability of a single permutation of $n$ objects. This is clearly intractable when we need to rank a large number of objects in real applications.

The *Luce model* is a stagewise model, which decomposes the process of generating a permutation of $n$ objects into $n$ sequential stages. At the $k$-th stage, an object is selected and assigned to position $k$

according to a probability based on the scores of the unassigned objects. The product of the selection probabilities at all the stages defines the probability of the permutation. The Luce model is highly efficient (with a polynomial time complexity) due to the decomposition. The expressiveness of the Luce model, however, is limited because it is defined on the scores of individual objects and cannot leverage versatile distance measures between permutations.

In this paper, we propose a new probabilistic model on permutations, which inherits the advantages of both the Luce model and the Mallows model and avoids their limitations. We refer to the model as coset-permutation distance based stagewise (CPS) model. Different from the Mallows model, the CPS model is a stagewise model. It decomposes the generative process of a permutation $\pi$ into sequential stages, which makes the efficient computation possible. At the $k$-th stage, an object is selected and assigned to position $k$ with a certain probability. Different from the Luce model, the CPS model defines the selection probability based on the distance between a location permutation $\sigma$ and the right coset of $\pi$ (referred to as coset-permutation distance) at each stage. In this sense, it is also a distance-based model. Because many different permutation distances can be used to induce the coset-permutation distance, the CPS model also has rich expressiveness. Furthermore, the coset-permutation distances induced by many popular permutation distances can be computed with polynomial time complexity, which further ensures the efficiency of the CPS model.

We then apply the CPS model to supervised rank aggregation and derive corresponding algorithms for learning and inference of the model. Experiments on public datasets show that the CPS model based algorithms can achieve state-of-the-art ranking accuracy, and are much more efficient than baseline methods based on previous probabilistic models.

## 2 Background

### 2.1 Rank Aggregation

There are mainly two kinds of rank aggregation, i.e., score-based rank aggregation [17, 16] and order-based rank aggregation [2, 7, 3]. In the former, objects in the input rankings are associated with scores, while in the latter, only the order information of these objects is available. In this work, we focus on the order-based rank aggregation, because it is more popular in real applications [7], and score-based rank aggregation can be easily converted to order-based rank aggregation by ignoring the additional score information [7].

Early methods for rank aggregation are heuristic based. For example, BordaCount [2, 7] and median rank aggregation [8] are simply based on average rank positions or the number of pairwise wins. In the recent literature, probabilistic models on permutations, such as the Mallows model and the Luce model, have been introduced to solve the problem of rank aggregation. Previous studies have shown that the probabilistic model based algorithms can outperform the heuristic methods in many settings. For example, the Mallows model has been shown very effective in both supervised rank aggregation and unsupervised rank aggregation, and the effectiveness of the Luce model has been demonstrated in the context of unsupervised rank aggregation. In the next subsection, we will describe these two models in more detail.

### 2.2 Probabilistic Models on Permutations

In order to better illustrate the probabilistic models on permutations, we first introduce some concepts and notations.

Let $\{1, 2, \ldots, n\}$ be a set of objects to be ranked. A ranking/permutation[1] $\pi$ is a bijection from $\{1, 2, \ldots, n\}$ to itself. We use $\pi(i)$ to denote the position given to object $i$ and $\pi^{-1}(i)$ to denote the object assigned to position $i$. We usually write $\pi$ and $\pi^{-1}$ as vectors whose $i$-th component is $\pi(i)$ and $\pi^{-1}(i)$, respectively. We also use the bracket alternative notation to represent a permutation, i.e., $\pi = \langle \pi^{-1}(1), \pi^{-1}(2), \ldots, \pi^{-1}(n) \rangle$.

The collection of all permutations of $n$ objects forms a non-abelian group under composition, called the symmetric group of order $n$, denoted as $S_n$. Let $S_{n-k}$ denote the subgroup of $S_n$ consisting of

all permutations whose first $k$ positions are fixed:

$$S_{n-k} = \{\pi \in S_n | \pi(i) = i, \forall i = 1, \ldots, k\}. \tag{1}$$

The right coset $S_{n-k}\pi = \{\sigma\pi | \sigma \in S_{n-k}\}$ is a subset of permutations whose top-$k$ objects are exactly the same as in $\pi$. In other words,

$$S_{n-k}\pi = \{\sigma | \sigma \in S_n, \sigma^{-1}(i) = \pi^{-1}(i), \forall i = 1, \ldots, k\}.$$

We also use $S_{n-k}(\langle i_1, i_2, \ldots, i_k \rangle)$ to denote the right coset with object $i_1$ in position 1, $i_2$ in position 2, $\ldots$, and $i_k$ in position $k$.

The Mallows model is a distance based probabilistic model on permutations. It uses a permutation distance $d$ on the symmetric group $S_n$ to define the probability of a permutation:

$$P(\pi|\theta, \sigma) = \frac{1}{Z(\theta, \sigma)} \exp(-\theta d(\pi, \sigma)), \tag{2}$$

where $\sigma \in S_n$ is the location permutation, $\theta \in \mathcal{R}$ is a dispersion parameter, and

$$Z(\theta, \sigma) = \sum_{\pi \in S_n} \exp(-\theta d(\pi, \sigma)). \tag{3}$$

There are many well-defined metrics to measure the distance between two permutations, such as Spearman's rank correlation $d_r(\pi, \sigma) = \sum_{i=1}^n (\pi(i) - \sigma(i))^2$, Spearman's footrule $d_f(\pi, \sigma) = \sum_{i=1}^n |\pi(i) - \sigma(i)|$, and Kendall's tau $d_t(\pi, \sigma) = \sum_{i=1}^n \sum_{j>i} 1_{\{\pi\sigma^{-1}(i) > \pi\sigma^{-1}(j)\}}$, where $1_{\{x\}} = 1$ if $x$ is true and 0 otherwise. One can (and sometimes should) choose different distances for different applications. In this regard, the Mallows model has rich expressiveness.

Note that there are $n!$ permutations in $S_n$. The computation of $Z(\theta, \sigma)$ involves the sum of $n!$ items. Although for some specific distances (such as $d_t$), there exist efficient ways for parameter estimation in the Mallows model, for many other distances (such as $d_r$ and $d_f$), there is no known efficient method to compute $Z(\theta, \sigma)$ and one has to pay for the high computational complexity of $O(n!)$ [9]. This has greatly limited the application of the Mallows model in real problems. Usually, one has to employ sampling methods such as MCMC to reduce the complexity [12, 11]. This, however, will affect the effectiveness of the model.

The Luce model is a stagewise probabilistic model on permutations. It assumes that there is a (hidden) score $\omega_i, i = 1, \ldots, n$, for each individual object $i$. To generate a permutation $\pi$, firstly the object $\pi^{-1}(1)$ is assigned to position 1 with probability $\frac{\exp(\omega_{\pi^{-1}(1)})}{\sum_{i=1}^n \exp(\omega_{\pi^{-1}(i)})}$; secondly the object $\pi^{-1}(2)$ is assigned to position 2 with probability $\frac{\exp(\omega_{\pi^{-1}(2)})}{\sum_{i=2}^n \exp(\omega_{\pi^{-1}(i)})}$; the assignment is continued until a complete permutation is formed. In this way, we obtain the permutation probability of $\pi$ as follows,

$$P(\pi) = \prod_{i=1}^n \frac{\exp(\omega_{\pi^{-1}(i)})}{\sum_{j=i}^n \exp(\omega_{\pi^{-1}(j)})}. \tag{4}$$

The computation of permutation probability in the Luce model is very efficient, as shown above. Actually the corresponding complexity is in the polynomial order of the number of objects. This is a clear advantage over the Mallows model. However, the Luce model is defined as a specific function of the scores of the objects, and therefore cannot make use of versatile permutation distances. As a result, its expressiveness is not as rich as the Mallows model, which may limit its applications.

## 3   A New Probabilistic Model

As discussed in the above section, both the Mallows and the Luce model have certain advantages and limitations. In this section, we propose a new probabilistic model on permutations, which can inherit their advantages and avoid their limitations. We call this model the coset-permutation distance based stagewise (CPS) model.

### 3.1 The CPS Model

As indicated by the name, the CPS model is defined on the basis of the so-called coset-permutation distance. A coset-permutation distance is induced from a permutation distance, as shown in the following definition.

**Definition 1.** *Given a permutation distance $d$, the **coset-permutation distance** $\hat{d}$ from a coset $S_{n-k}\pi$ to a target permutation $\sigma$ is defined as the average distance between the permutations in the coset and the target permutation:*

$$\hat{d}(S_{n-k}\pi, \sigma) = \frac{1}{|S_{n-k}\pi|} \sum_{\tau \in S_{n-k}\pi} d(\tau, \sigma), \tag{5}$$

*where $|S_{n-k}\pi|$ is the number of permutations in set $S_{n-k}\pi$.*

It is easy to verify that if the permutation distance $d$ is right invariant, then the induced coset-permutation distance $\hat{d}$ is also right invariant.

With the concept of coset-permutation distance, given a dispersion parameter $\theta \in \mathcal{R}$ and a location permutation $\sigma \in S_n$, we can define the CPS model as follows. Specifically, the generative process of a permutation $\pi$ of $n$ objects is decomposed into $n$ sequential stages. As an initialization, all the objects are placed in a working set. At the $k$-th stage, the task is to select the $k$-th object in the original permutation $\pi$ out of the working set. The probability of this selection is defined with the coset-permutation distance between the right coset $S_{n-k}\pi$ and the location permutation $\sigma$:

$$\frac{\exp(-\theta\hat{d}(S_{n-k}\pi, \sigma))}{\sum_{j=k}^{n} \exp(-\theta\hat{d}(S_{n-k}(\pi, k, j), \sigma))}, \tag{6}$$

where $S_{n-k}(\pi, k, j)$ denotes the right coset including all the permutations that rank objects $\pi^{-1}(1), \ldots, \pi^{-1}(k-1)$, and $\pi^{-1}(j)$ in the top $k$ positions respectively.

From Eq. (6), we can see that the closer the coset $S_{n-k}\pi$ is to the location permutation $\sigma$, the larger the selection probability is. Considering all the $n$ stages, we will obtain the overall probability of generating $\pi$, which is shown in the following definition.

**Definition 2.** *The **CPS model** defines the probability of a permutation $\pi$ conditioned on a dispersion parameter $\theta$ and a location permutation $\sigma$ as:*

$$P(\pi|\theta, \sigma) = \prod_{k=1}^{n} \frac{\exp(-\theta\hat{d}(S_{n-k}\pi, \sigma))}{\sum_{j=k}^{n} \exp(-\theta\hat{d}(S_{n-k}(\pi, k, j), \sigma))}, \tag{7}$$

*where $S_{n-k}(\pi, k, j)$ is defined in the sentence after Eq. (6).*

It is easy to verify that the probabilities $P(\pi|\theta, \sigma), \pi \in S_n$ defined in the CPS model naturally form a distribution over $S_n$. That is, for each $\pi \in S_n$, we always have $P(\pi|\theta, \sigma) \geq 0$, and $\sum_{\pi \in S_n} P(\pi|\theta, \sigma) = 1$.

In rank aggregation, one usually needs to combine multiple input rankings. To deal with this scenario, we further extend the CPS model, following the methodology used in [12].

$$P(\pi|\boldsymbol{\theta}, \boldsymbol{\sigma}) = \prod_{i=1}^{n} \frac{e^{-\sum_{m=1}^{M} \theta_m \hat{d}(S_{n-i}\pi, \sigma_m)}}{\sum_{j=i}^{n} e^{-\sum_{m=1}^{M} \theta_m \hat{d}(S_{n-i}(\pi, i, j), \sigma_m)}}, \tag{8}$$

where $\boldsymbol{\theta} = \{\theta_1, \ldots, \theta_M\}$ and $\boldsymbol{\sigma} = \{\sigma_1, \ldots, \sigma_M\}$.

The CPS model defined as above can be computed in a highly efficient manner, as discussed in the following subsection.

### 3.2 Computational Complexity

According to the definition of the CPS model, at the $k$-th stage, one needs to compute $(n - k)$ coset-permutation distances. At first glance, the complexity of computing each coset-permutation

distance is about $O((n-k)!)$, since the coset contains this number of permutations. This is clearly intractable. The good news is that the real complexity for computing the coset-permutation distance induced by several popular permutation distances is much lower than $O((n-k)!)$. Actually, they can be as low as $O(n^2)$, according to the following theorem.

**Theorem 1.** *The coset-permutation distances induced from Spearman's rank correlation $d_r$, Spearman's footrule $d_f$, and Kendall's tau $d_t$ can all be computed with a complexity of $O(n^2)$. More specifically, for $k = 1, 2, \ldots, n - 2$, we have*[2]

$$\hat{d}_r(S_{n-k}\pi, \sigma) = \sum_{i=1}^{k} (\sigma(\pi^{-1}(i)) - i)^2 + \frac{1}{n-k} \sum_{i=k+1}^{n} \sum_{j=k+1}^{n} (\sigma(\pi^{-1}(i)) - j)^2, \tag{9}$$

$$\hat{d}_f(S_{n-k}\pi, \sigma) = \sum_{i=1}^{k} |\sigma(\pi^{-1}(i)) - i| + \frac{1}{n-k} \sum_{i=k+1}^{n} \sum_{j=k+1}^{n} |\sigma(\pi^{-1}(i)) - j|, \tag{10}$$

$$\hat{d}_t(S_{n-k}\pi, \sigma) = \frac{1}{4}(n-k)(n-k-1) + \sum_{i=1}^{k} \sum_{j=i+1}^{n} 1_{\{\sigma(\pi^{-1}(i)) > \sigma(\pi^{-1}(j))\}}. \tag{11}$$

According to the above theorem, each induced coset-permutation distance can be computed with a time complexity of $O(n^2)$. If we compute the CPS model according to Eq. (7), the time complexity will then be $O(n^4)$. This is clearly much more efficient than $O((n-k)!)$. Moreover, with careful implementations, the time complexity of $O(n^4)$ can be further reduced to $O(n^2)$, as indicated by the following theorem.

**Theorem 2.** *For the coset distances induced from $d_r$, $d_f$ and $d_t$, the CPS model in Eq. (7) can be computed with a time complexity of $O(n^2)$.*

### 3.3 Relationship with Previous Models

The CPS model as defined above has strong connections with both the Luce model and the Mallows model, as shown below.

The similarity between the CPS model and the Luce model is that they are both defined in a stagewise manner. This stagewise definition enables efficient inference for both models. The difference between the CPS model and the Luce model lies in that the CPS model has a much richer expressiveness than the Luce model. This is mainly because the CPS model is a distance based model while the Luce model is not. Our experiments in Section 5 show that different distances may be appropriate for different applications and datasets, which means a model with rich expressiveness has the potential to be applied for versatile applications.

The similarity between the CPS model and the Mallows model is that they are both based on distances. Actually when the coset-permutation distance in the CPS model is induced by the Kendall's tau $d_t$, the CPS model is even mathematically equivalent to the Mallows model defined with $d_t$. The major difference between the CPS model and the Mallows model lies in the computational efficiency. The CPS model can be computed efficiently with a polynomial time complexity, as discussed in the previous sub section. However, for most permutation distances, the complexity of the Mallows model is as huge as $O(n!)$.[3]

According to the above discussions, we can see that the CPS model inherits the advantages of both the Luce model and the Mallows model, and avoids their limitations.

## 4 Algorithms for Rank Aggregation

In this section, we show how to apply the extended CPS model to solve the problem of rank aggregation. Here we take meta search as an example, and consider the supervised case of rank aggregation. That is, given a set of training queries, we need to learn the parameters $\boldsymbol{\theta}$ in the CPS model and apply the model with the learned parameters to aggregate rankings for new test queries.

**Algorithm 1** Sequential inference

---

**Input:** parameters $\boldsymbol{\theta}$, input rankings $\boldsymbol{\sigma}$

**Inference:**

1:    Initialize the set of $n$ objects: $D = \{1, 2, \ldots, n\}$.

2:    $\pi^{-1}(1) = \arg\min_{j \in D} \sum_m \theta_m \hat{d}(S_{n-1}(<j>), \sigma_m)$.

3:    Remove object $\pi^{-1}(1)$ from set $D$.

4:    for $k = 2$ to $n$

     (4.1): $\pi^{-1}(k) = \arg\min_{j \in D} \sum_m \theta_m \hat{d}\left(S_{n-k}(<\pi^{-1}(1), \ldots, \pi^{-1}(k-1), j>), \sigma_m\right)$,

     (4.2): Remove object $\pi^{-1}(k)$ from set $D$.

5:    end

**Output:** the final ranking $\pi$.

---

## 4.1   Learning

Let $D = \{(\pi^{(l)}, \boldsymbol{\sigma}^{(l)})\}$ be the set of training queries, in which $\pi^{(l)}$ is the ground truth ranking for query $q_l$, and $\boldsymbol{\sigma}^{(l)}$ is the set of $M$ input rankings.

In order to learn the parameters $\boldsymbol{\theta}$ in Eq. (8), we employ maximum likelihood estimation. Specifically, the log likelihood of the training data for the CPS model can be written as below,

$$
\begin{aligned}
L(\boldsymbol{\theta}) &= \log \prod_l P(\pi^{(l)}|\boldsymbol{\theta}, \boldsymbol{\sigma}^{(l)}) = \sum_l \log P(\pi^{(l)}|\boldsymbol{\theta}, \boldsymbol{\sigma}^{(l)}) \\
&= \sum_l \sum_{k=1}^{n} \left\{ -\sum_{m=1}^{M} \theta_m \hat{d}(S_{n-k}\pi^{(l)}, \sigma_m^{(l)}) - \log \sum_{j=k}^{n} e^{-\sum_{m=1}^{M} \theta_m \hat{d}(S_{n-k}(\pi^{(l)}, k, j), \sigma_m^{(l)})} \right\}
\end{aligned} \quad (12)
$$

It is not difficult to prove that $L(\boldsymbol{\theta})$ is concave with respect to $\boldsymbol{\theta}$. Therefore, we can use simple optimization techniques like gradient ascent to find the globally optimal $\boldsymbol{\theta}$.

## 4.2   Inference

In the test phase, given a new query and its associated $M$ input rankings, we need to infer a final ranking with the learned parameters $\boldsymbol{\theta}$.

A straightforward method is to find the permutation with the largest probability conditioned on the $M$ input rankings, just as the widely-used inference algorithm for the Mallows model [12]. We call the method global inference since it finds the globally most likely one from all possible permutations.

The problem with global inference lies in that its complexity is as high as $O(n!)$. As a consequence, it cannot handle applications with a large number of objects to rank. Considering the stagewise definition of the CPS model, we propose a sequential inference algorithm. The algorithm decomposes the inference into $n$ steps. At the $k$-th step, we select the object $j$ that can minimize the coset-permutation distance $\sum_m \theta_m \hat{d}(S_{n-k}(\langle \pi^{-1}(1), \ldots, \pi^{-1}(k-1), j \rangle), \sigma_m)$, and put it at the $k$-th position. The procedure is listed in Algorithm 1.

In fact, sequential inference is an approximation of global inference, with a much lower complexity. Theorem 3 shows that the complexity of sequential inference is just $O(Mn^2)$. Our experiments in the next section indicate that such an approximation does not hurt the ranking accuracy by much, while significantly speeds up the inference process.

**Theorem 3.** *For the coset distance induced from $d_r$, $d_f$, and $d_t$, the stagewise inference as shown in Algorithm 1 can be conducted with a time complexity of $O(Mn^2)$.*

# 5   Experimental Results

We have performed experiments to test the efficiency and effectiveness of the proposed CPS model.

## 5.1 Settings

We take meta search as the target application, and use the LETOR [13] benchmark datasets in the experiments. LETOR is a public collection created for ranking research.[4] There are two meta search datasets in LETOR, MQ2007-agg and MQ2008-agg. In addition to using them, we also composed a smaller dataset from MQ2008-agg, referred to as MQ2008-small, by selecting queries with no more than 8 documents from the MQ2008-agg dataset. This small dataset is used to perform detailed investigations on the CPS model and other baseline models.

There are three levels of relevance labels in all the datasets: highly relevant, relevant, and irrelevant. We used NDCG [10] as the evaluation measure in our experiments. NDCG is a widely-used IR measure for multi-level relevance judgments. The larger the NDCG value, the better the aggregation accuracy.

The 5-fold cross validation strategy was adopted for all the datasets. All the results reported in this section are the average results over the five folds.

For the CPS model, we tested two inference methods: global inference (denoted as CPS-G) and sequential inference (denoted as CPS-S). For comparison, we implemented the Mallows model. When applied to supervised rank aggregation, the learning process of the Mallows model is also maximum likelihood estimation. For inference, we chose the permutation with the maximal probability as the final aggregated ranking. The time complexity of both learning and inference of the Mallows model with distance $d_r$ and $d_f$ is $O(n!)$. We also implemented an approximate algorithm as suggested by [12] using MCMC sampling to speed up the learning process. We refer to this approximate algorithm as MallApp. Note that the time complexity of the inference of MallApp is still $O(n!)$ for distance $d_r$ and $d_f$. Furthermore, as a reference, we also tested a traditional method, BordaCount [1], which is based on majority voting. We did not compare with the Luce model because it is not straightforward to be applied to supervised rank aggregation, as far as we know.

Note that Mallows, MallApp and CPS-G cannot handle the large datasets MQ2007-agg and MQ2008-agg, and were only tested on the small dataset MQ2008-small.

## 5.2 Results

First, we report the results of these algorithms on the MQ2008-small dataset.

The aggregation accuracies in terms of NDCG are listed in Table 1(a). Note that the accuracy of Mallows($d_t$) is the same as that of CPS-G($d_t$) because of the mathematical equivalence of the two models. Therefore, we omit Mallows($d_t$) in the table. We did not implement the sampling-based learning algorithm for the Mallows model with distance $d_t$, because in this case the learning algorithm has already been efficient enough.

From the table, we have the following observations.

- For the Mallows model, exact learning is a little better than the approximate learning, especially for distance $d_f$. This is in accordance with our intuition. Sampling can improve the efficiency of the algorithm, but also miss some information contained in the original permutation probability space.

- For the CPS model, the sequential inference does not lead to much accuracy drop as compared to global inference. For distances $d_f$ and $d_r$, the CPS model outperforms the Mallows model. For example, when $d_f$ is used, the CPS model wins the Mallows model by about 0.04 in terms of NDCG@2, which corresponds to a relative improvement of 10%.

- For the same model, with different distance functions, the performances differ significantly. This indicates that one should select the most suitable distance for a given application.

- All the probabilistic model based methods are better than BordaCount, the heuristic method.

In addition to the comparison of aggregation accuracy, we have also logged the running time of each model. For example, on our test machine (with 2.13Ghz CPU and 4GB memory), it took about 12

Table 1: Results

(a) Results on MQ2008-small

| NDCG | @2 | @4 | @6 | @8 |
|---|---|---|---|---|
| BordaCount | 0.335 | 0.421 | 0.479 | 0.420 |
| CPS-G($d_f$) | 0.392 | 0.471 | 0.518 | 0.446 |
| CPS-S($d_f$) | 0.389 | 0.471 | 0.517 | 0.444 |
| Mallows($d_f$) | 0.350 | 0.449 | 0.490 | 0.422 |
| MallApp($d_f$) | 0.343 | 0.440 | 0.491 | 0.420 |
| CPS-G($d_r$) | 0.387 | 0.476 | 0.519 | 0.443 |
| CPS-S($d_r$) | 0.388 | 0.478 | 0.519 | 0.441 |
| Mallows($d_r$) | 0.333 | 0.442 | 0.491 | 0.420 |
| MallApp($d_r$) | 0.343 | 0.440 | 0.490 | 0.419 |
| CPS-G($d_t$) | 0.414 | 0.485 | 0.530 | 0.451 |
| CPS-S($d_t$) | 0.419 | 0.489 | 0.534 | 0.454 |

(b) Results on MQ2008-agg and MQ2007-agg

| on MQ2008-agg | | | | |
|---|---|---|---|---|
| NDCG | @2 | @4 | @6 | @8 |
| BordaCount | 0.281 | 0.343 | 0.389 | 0.372 |
| CPS-S($d_t$) | 0.312 | 0.379 | 0.420 | 0.403 |
| CPS-S($d_r$) | 0.314 | 0.376 | 0.419 | 0.398 |
| CPS-S($d_f$) | 0.276 | 0.352 | 0.399 | 0.383 |
| on MQ2007-agg | | | | |
| NDCG | @2 | @4 | @6 | @8 |
| BordaCount | 0.201 | 0.213 | 0.225 | 0.238 |
| CPS-S($d_t$) | 0.298 | 0.311 | 0.322 | 0.335 |
| CPS-S($d_r$) | 0.332 | 0.341 | 0.352 | 0.362 |
| CPS-S($d_f$) | 0.298 | 0.312 | 0.323 | 0.336 |

seconds for CPS-G($d_f$),[5] 30 seconds for MallApp($d_f$), and 12 hours for Mallows($d_f$) to finish the training process. The inference of the Mallows model based algorithms and the global inference of the CPS model based algorithms took more time than sequential inference of the CPS model, although the difference was not significant (this is mainly because $n \leq 8$ for MQ2008-small).

From these results, we can see that the proposed CPS model plus sequential inference is the most efficient one, and its accuracy is also very good as compared to other methods.

Second, we report the results on MQ2008-agg and MQ2007-agg in Table 1(b). Note that the results of the Mallows model based algorithms and that of the CPS model with global inference are not available because of the high computational complexity for their learning or inference. The results show that the CPS model with sequential inference outperforms BordaCount, no matter which distance is used. Moreover, the CPS model with $d_t$ performs the best on MQ2008-agg, and the model with $d_r$ performs the best on MQ2007-agg. This indicates that we can achieve good ranking performance by choosing the most suitable distances for different datasets (and so applications). This provides a side evidence that it is beneficial for a probabilistic model on permutations to have rich expressiveness.

To sum up, the experimental results indicate that the CPS model based learning and sequential inference algorithms can achieve state-of-the-art ranking accuracy and are more efficient than other algorithms.

# 6  Conclusions and Future Work

In this paper, we have proposed a new probabilistic model, named the CPS model, on permutations for rank aggregation. The model is based on coset-permutation distance and defined in a stagewise manner. It inherits the advantages of the Luce model (high efficiency) and the Mallows model (rich expressiveness), and avoids their limitations. We have applied the model to supervised rank aggregation and investigated how to perform learning and inference. Experiments on public datasets demonstrate the effectiveness and efficiency of the CPS model.

As future work, we plan to investigate the following issues. (1) We have shown that three induced coset-permutation distances can be computed efficiently. We will explore whether other distances also have such properties. (2) We have applied the CPS model to the supervised case of rank aggregation. We will study the unsupervised case. (3) We will investigate other applications of the model, and discuss how to select the most suitable distance for a given application.

## Footnotes

[1]We will interchangeably use the two terms in the paper.

[2]Note that $\hat{d}(S_{n-k}\pi, \sigma) = d(\pi, \sigma)$ for $k = n - 1, n$.

[3]An exception is that for Kendall's tau distance, the Mallows model can be as efficient as the CPS model because they are mathematically equivalent.

[4]The datasets can be downloaded from `http://research.microsoft.com/~letor`.

[5]The training process of CPS-G and CPS-S is exactly the same.

# References

[1] J. Aslam and M. Montague. Models for metasearch. In *Proceedings of the 24th SIGIR*, pages 276–284, 2001.

[2] J. A. Aslam and M. Montague. Models for metasearch. In *SIGIR '01: Proceedings of the 24th annual international ACM SIGIR conference on Research and development in information retrieval*, pages 276–284, New York, NY, USA, 2001. ACM.

[3] M. Beg. Parallel Rank Aggregation for the World Wide Web. *World Wide Web. Kluwer Academic Publishers*, 6(1):5–22, 2004.

[4] D. Critchlow. *Metric methods for analyzing partially ranked data*. 1980.

[5] H. Daniels. Rank correlation and population models. *Journal of the Royal Statistical Society. Series B (Methodological)*, pages 171–191, 1950.

[6] P. Diaconis. *Group representations in probability and statistics*. Institute of Mathematical Statistics Hayward, CA, 1988.

[7] C. Dwork, R. Kumar, M. Naor, and D. Sivakumar. Rank aggregation methods for the web. In *WWW '01: Proceedings of the 10th international conference on World Wide Web*, pages 613–622, New York, NY, USA, 2001. ACM.

[8] R. Fagin, R. Kumar, and D. Sivakumar. Efficient similarity search and classification via rank aggregation. In *SIGMOD '03: Proceedings of the 2003 ACM SIGMOD international conference on Management of data*, pages 301–312, New York, NY, USA, 2003. ACM.

[9] M. Fligner and J. Verducci. Distance based ranking models. *Journal of the Royal Statistical Society. Series B (Methodological)*, 48(3):359–369, 1986.

[10] J. Kalervo and K. Kekäläinen. Cumulated gain-based evaluation of ir techniques. *ACM Trans. Inf. Syst.*, 20(4):422–446, 2002.

[11] A. Klementiev, D. Roth, and K. Small. Unsupervised rank aggregation with distance-based models. In *Proceedings of the 25th ICML*, pages 472–479, 2008.

[12] G. Lebanon and J. Lafferty. Cranking: Combining rankings using conditional probability models on permutations. In *ICML2002*, pages 363–370, 2002.

[13] T. Liu, J. Xu, T. Qin, W. Xiong, and H. Li. LETOR: Benchmark dataset for research on learning to rank for information retrieval. In *SIGIR2007 Workshop on Learning to Rank for Information Retrieval*, pages 3–10, 2007.

[14] R. D. Luce. *Individual Choice Behavior*. Wiley, 1959.

[15] C. L. Mallows. Non-null ranking models. *Biometrika*, 44:114–130, 1957.

[16] R. Manmatha, T. Rath, and F. Feng. Modeling score distributions for combining the outputs of search engines. In *SIGIR '01: Proceedings of the 24th annual international ACM SIGIR conference on Research and development in information retrieval*, pages 267–275, New York, NY, USA, 2001. ACM.

[17] M. Montague and J. A. Aslam. Relevance score normalization for metasearch. In *CIKM '01: Proceedings of the tenth international conference on Information and knowledge management*, pages 427–433, New York, NY, USA, 2001. ACM.

[18] R. L. Plackett. The analysis of permutations. *Applied Statistics*, 24(2):193–202, 1975.

[19] L. Thurstone. A law of comparative judgment. *Psychological review*, 34(4):273–286, 1927.

